# Gaussian Process Models for Link Analysis and Transfer Learning

**Kai Yu**
NEC Laboratories America
Cupertino, CA 95014

**Wei Chu**
Columbia University, CCLS
New York, NY 10115

## Abstract

This paper aims to model relational data on edges of networks. We describe appropriate Gaussian Processes (GPs) for directed, undirected, and bipartite networks. The inter-dependencies of edges can be effectively modeled by adapting the GP hyper-parameters. The framework suggests an intimate connection between link prediction and transfer learning, which were traditionally two separate research topics. We develop an efficient learning algorithm that can handle a large number of observations. The experimental results on several real-world data sets verify superior learning capacity.

## 1 Introduction

In many scenarios the data of interest consist of relational observations on the edges of networks. Typically, a given finite collection of such relational data can be represented as an $M \times N$ matrix $\mathbf{Y} = \{y_{i,j}\}$, which is often partially observed because many elements are missing. Sometimes accompanying $\mathbf{Y}$ are attributes of nodes or edges. As an important nature of networks, $\{y_{i,j}\}$ are highly inter-dependent even conditioned on known node or edge attributes. The phenomenon is extremely common in real-world data, for example,

- *Bipartite Graphs*. The data represent relations between two different sets of objects or measurements under a pair of heterogeneous conditions. One notable example is *transfer learning*, also known as multi-task learning, which jointly learns multiple related but different predictive functions based on the $M \times N$ observed labels $\mathbf{Y}$, namely, the results of $N$ functions acting on a set of $M$ data examples. Collaborative filtering is an important application of transfer learning that learns many users' interests on a large set of items.

- *Undirected and Directed Graphs*. The data are measurements of existences, strengths, and types of links between a set of nodes in a graph, where a given collection of observations are an $M \times M$ (in this case $N = M$) matrix $\mathbf{Y}$, which can be symmetric or asymmetric, depending on whether the links are undirected or directed. Examples include protein-protein interactions, social networks, citation networks, and hyperlinks on the WEB. Link prediction aims to recover those missing measurements in $\mathbf{Y}$, for example, predicting unknown protein-protein interactions based on known interactions.

The goal of this paper is to design a Gaussian process (GP) [13] framework to model the dependence structure of networks, and to contribute an efficient algorithm to learn and predict large-scale relational data. We explicitly construct a series of parametric models indexed by their dimensionality, and show that in the limit we obtain nonparametric GP priors consistent with the dependence of edge-wise measurements. Since the kernel matrix is on a quadratic number of edges and the computation cost is even cubic of the kernel size, we develop an efficient algorithm to reduce the computational complexity. We also demonstrate that transfer learning has an intimate connection to link prediction. Our method generalizes several recent transfer learning algorithms by additionally learning a task-specific kernel that directly expresses the dependence between tasks.

The application of GPs to learning on networks or graphs has been fairly recent. Most of the work in this direction has focused on GPs over nodes of graphs and targeted at the classification of nodes [20, 6, 10]. In this paper, we regard the *edges* as the first-class citizen and develop a general GP framework for modeling the dependence of edge-wise observations on bipartite, undirected and directed graphs. This work extends [19], which built GPs for only bipartite graphs and proposed an algorithm scaling cubically to the number of nodes. In contrast, the work here is more general and the algorithm scales *linearly* to the number of edges. Our study promises a careful treatment to model the nature of edge-wise observations and offers a promising tool for link prediction.

## 2 Gaussian Processes for Network Data

### 2.1 Modeling Bipartite Graphs

We first review the edge-wise GP for bipartite graphs [19], where each observation is a measurement on a pair of objects of different types, or under a pair of heterogenous conditions. Formally, let $\mathcal{U}$ and $\mathcal{V}$ be two index sets, then $y_{i,j}$ denotes a measurement on edge $(i, j)$ with $i \in \mathcal{U}$ and $j \in \mathcal{V}$. In the context of transfer learning, the pair involves a data instance $i$ and a task $j$, and $y_{i,j}$ denotes the label of data $i$ within task $j$. The probabilistic model assumes that $y_{i,j}$ are noisy outcomes of a real-valued function $f : \mathcal{U} \times \mathcal{V} \to \mathbb{R}$, which follows a Gaussian process $\mathcal{GP}(b, K)$, characterized by mean function $b$ and covariance (kernel) function between edges

$$K\left((i, j), (i', j')\right) = \Sigma(i, i')\Omega(j, j') \tag{1}$$

where $\Sigma$ and $\Omega$ are kernel functions on $\mathcal{U}$ and $\mathcal{V}$, respectively. As a result, the realizations of $f$ on a finite set $i = 1, \ldots, M$ and $j = 1, \ldots, N$ form a matrix $\mathbf{F}$, following a matrix-variate normal distribution $\mathcal{N}_{M \times N}(\mathbf{B}, \mathbf{\Sigma}, \mathbf{\Omega})$, or equivalently a normal distribution $\mathcal{N}(\mathbf{b}, \mathbf{K})$ with mean $\mathbf{b} = \text{vec}(\mathbf{B})$ and covariance $\mathbf{K} = \mathbf{\Omega} \otimes \mathbf{\Sigma}$, where $\otimes$ means Kronecker product. The dependence structure of edges is decomposed into the dependence of nodes. Since a kernel is a notion of similarity, the model expresses a prior belief – if node $i$ is similar to node $i'$ and node $j$ is similar node $j'$, then so are $f(i, j)$ and $f(i', j')$.

It is essential to learn the kernels $\Sigma$ and $\Omega$ based on the partially observed $\mathbf{Y}$, in order to capture the dependence structure of the network. For transfer learning, this means to learn the kernel $\Sigma$ between data instances and the kernel $\Omega$ between tasks. Having $\Sigma$ and $\Omega$ is it then possible to predict those missing $y_{i,j}$ based on known observations by using GP inference.

**Theorem 2.1** ([19])**.** *Let* $f(i, j) = D^{-1/2} \sum_{k=1}^{D} g_k(i)h_k(j) + b(i, j)$, *where* $g_k \overset{iid}{\sim} \mathcal{GP}(0, \Sigma)$ *and* $h_k \overset{iid}{\sim} \mathcal{GP}(0, \Omega)$, *then* $f \sim \mathcal{GP}(b, K)$ *in the limit* $D \to \infty$, *and the covariance between pairs is* $K\left((i, j), (i', j')\right) = \Sigma(i, i')\Omega(j, j')$.

Theorem (2.1) offers an alternative view to understand the model. The *edge-wise* function $f$ can be decomposed into a product of two sets of intermediate *node-wise* functions, $\{g_k\}_{k=1}^{\infty}$ and $\{h_k\}_{k=1}^{\infty}$, which are i.i.d. samples from two GP priors $\mathcal{GP}(0, \Sigma)$ and $\mathcal{GP}(0, \Omega)$. The theorem suggests that the GP model for bipartite relational data is a generalization of a Bayesian low-rank matrix factorization $\mathbf{F} = \mathbf{H}\mathbf{G}^\top + \mathbf{B}$, under the prior $\mathbf{H} \sim \mathcal{N}_{M \times D}(0, \mathbf{\Sigma}, \mathbf{I})$ and $\mathbf{G} \sim \mathcal{N}_{N \times D}(0, \mathbf{\Omega}, \mathbf{I})$. When $D$ is finite, the elements of $\mathbf{F}$ are not Gaussian random variables.

### 2.2 Modeling Directed and Undirected Graphs

In this section we model observations on pairs of nodes of the *same* set $\mathcal{U}$. This case includes both directed and undirected graphs. It turns out that the directed graph is relatively easy to handle while deriving a GP prior for undirected graphs is slightly non-trivial. For the case of directed graphs, we let the function $f : \mathcal{U} \times \mathcal{U} \to \mathbb{R}$ follow $\mathcal{GP}(b, K)$, where the covariance function between edges is

$$K\left((i, j), (i', j')\right) = C(i, i')C(j, j') \tag{2}$$

and $C : \mathcal{U} \times \mathcal{U} \to \mathbb{R}$ is a kernel function between nodes. Since a random function $f$ drawn from the GP is generally *asymmetric* (even if $b$ is symmetric), namely $f(i, j) \neq f(j, i)$, the direction of edges can be modeled. The covariance function Eq. (2) can be derived from Theorem (2.1) by setting that $\{g_k\}$ and $\{h_k\}$ are two independent sets of functions i.i.d. sampled from the *same* GP prior

$\mathcal{GP}(0, C)$, modeling the situation that each node's behavior as a sender is different but statistically related to it's behavior as a receiver. This is a reasonable modeling assumption. For example, if two papers cite a common set of papers, their are also likely to be cited by a common set of other papers.

For the case of undirected graphs, we need to design a GP that ensures any sampled function to be *symmetric*. Following the construction of GP in Theorem (2.1), it seems that $f$ is symmetric if $g_k \equiv h_k$ for $k = 1, \ldots, D$. However a calculation reveals that $f$ is not bounded in the limit $D \to \infty$. Theorem (2.2) shows that the problem can be solved by subtracting a growing quantity $D^{1/2}C(i, j)$ as $D \to \infty$, and suggests the covariance function

$$K((i, j), (i', j')) = C(i, i')C(j, j') + C(i, j')C(j, i'). \tag{3}$$

With such covariance function , $f$ is ensured to be symmetric because the covariance between $f(i, j)$ and $f(j, i)$ equals the variance of either.

**Theorem 2.2.** *Let $f(i, j) = D^{-1/2} \sum_{k=1}^{D} t_k(i)t_k(j) + b(i, j) - D^{1/2}C(i, j)$, where $t_k \overset{iid}{\sim} \mathcal{GP}(0, C)$, then $f \sim \mathcal{GP}(b, K)$ in the limit $D \to \infty$, and the covariance between pairs is $K((i, j), (i', j')) = C(i, i')C(j, j') + C(i, j')C(j, i')$. If $b(i, j) = b(j, i)$, then $f(i, j) = f(j, i)$.*

*Proof.* Without loss of generality, let $b(i, j) \equiv 0$. Based on the central limit theorem, for every $(i, j)$, $f(i, j)$ converges to a zero-mean Gaussian random variable as $D \to \infty$, because $\{t_k(i)t_k(j)\}_{k=1}^{D}$ is a collection of random variables independently following the same distribution, and has the mean $C(i, j)$. The covariance function is $\text{Cov}(f(i, j), f(i', j')) = \frac{1}{D} \sum_{k=1}^{D} \{E[t_k(i)t_k(j)t_k(i')t_k(j')] - C(i, j)E[t_k(i')t_k(j')] - C(i', j')E[t_k(i)t_k(j)] + C(i, j)C(i', j')\} = C(i, i')C(j, j') + C(i, j')C(j, i') + C(i, j)C(i', j') - C(i, j)C(i', j') = C(i, i')C(j, j') + C(i, j')C(j, i')$. $\square$

Interestingly, Theorem (2.2) recovers Theorem (2.1) and is thus more general. To see the connection, let $h_k \sim \mathcal{GP}(0, \Sigma)$ and $g_k \sim \mathcal{GP}(0, \Omega)$ be concatenated to form a function $t_k$, then we have $t_k \sim \mathcal{GP}(0, C)$ and the covariance is

$$C(i, j) = \begin{cases} \Sigma(i, j), & \text{if } i, j \in \mathcal{U}, \\ \Omega(i, j), & \text{if } i, j \in \mathcal{V}, \\ 0, & \text{if } i, j \text{ are in different sets.} \end{cases} \tag{4}$$

For $i, i' \in \mathcal{U}$ and $j, j' \in \mathcal{V}$, applying Theorem (2.2) leads to

$$f(i, j) = D^{-1/2} \sum_{k=1}^{D} t_k(i)t_k(j) + b(i, j) - D^{1/2}C(i, j) = D^{-1/2} \sum_{k=1}^{D} h_k(i)g_k(j) + b(i, j), \tag{5}$$

$$K((i, j), (i', j')) = C(i, i')C(j, j') + C(i, j')C(j, i') = \Sigma(i, i')\Omega(j, j'). \tag{6}$$

Theorems (2.1) and (2.2) suggest a general GP framework to model directed or undirected relationships connecting heterogeneous types of nodes. Basically, we learn node-wise covariance functions, like $\Sigma$, $\Omega$, and $C$, such that edge-wise covariances composed by Eq. (1), (2), or (3) can explain the happening of observations $y_{i,j}$ on edges. The proposed framework can be extended to cope with more complex network data, for example, networks containing both undirected links and directed links. We will briefly discuss some extensions in Sec. 6.

## 3 An Efficient Learning Algorithm

We consider the regression case under a Gaussian noise model, and later briefly discuss extensions to the classification case. Let $\mathbf{y} = [y_{i,j}]_{(i,j) \in \mathbb{O}}$ be the observational vector of length $|\mathbb{O}|$, $\mathbf{f}$ be the corresponding quantities of the latent function $f$, and $\mathbf{K}$ be the $|\mathbb{O}| \times |\mathbb{O}|$ matrix of $K$ between edges having observations, computed by Eq. (1)-(3). Then observations on edges are generated by

$$y_{i,j} = f(i, j) + b_{i,j} + \epsilon_{i,j} \tag{7}$$

where $\mathbf{f} \sim \mathcal{N}(0, \mathbf{K})$, $\epsilon_{i,j} \overset{iid}{\sim} \mathcal{N}(0, \beta^{-1})$, and the mean has a parametric form $b_{i,j} = \mu_i + \nu_j$. In the directed/undirected graph case we let $\mu_i = \nu_i$ for any $i \in \mathcal{U}$. $\mathbf{f}$ can be analytically marginalized out, the marginal distribution of observations is then

$$p(\mathbf{y}|\theta) = \mathcal{N}(\mathbf{y}; \mathbf{b}, \mathbf{K} + \beta^{-1}\mathbf{I}), \tag{8}$$

where $\theta = \{\beta, b, K\}$. The parameters can be estimated by minimizing the penalized negative log-likelihood $\mathcal{L}(\theta) = -\ln p(\mathbf{y}|\theta) + \ell(\theta)$ under a suitable regularization $\ell(\theta)$. The objective function has the form:

$$\mathcal{L}(\theta) = \frac{|\mathbb{O}|}{2}\log 2\pi + \frac{1}{2}\ln|\mathbf{C}| + \frac{1}{2}\mathrm{tr}\left[\mathbf{C}^{-1}\mathbf{mm}^\top\right] + \ell(\theta), \tag{9}$$

where $\mathbf{C} = \mathbf{K} + \beta^{-1}\mathbf{I}$, $\mathbf{m} = \mathbf{y} - \mathbf{b}$ and $\mathbf{b} = [b_{i,j}]$, $(i,j) \in \mathbb{O}$. $\ell(\theta)$ will be configured in Sec. 3.1. Gradient-based optimization packages can be applied to find a local optimum of $\theta$. However the computation can be prohibitively high when the size $|\mathbb{O}|$ of measured edges is very big, because the memory cost is $\mathcal{O}(|\mathbb{O}|^2)$, and the computational cost is $\mathcal{O}(|\mathbb{O}|^3)$. In our experiments $|\mathbb{O}|$ is about tens of thousands or even millions. A slightly improved algorithm was introduced in [19], with a complexity $\mathcal{O}(M^3 + N^3)$ cubic to the size of nodes. The algorithm employed a non-Gaussian approximation based on Theorem (2.1) and is applicable to only bipartite graphs.

We reduce the memory and computational cost by exploring the special structure of $K$ as discussed in Sec. 2 and assume $K$ to be composed by node-wise linear kernels $\Sigma(i, i') = \langle \mathbf{x}_i, \mathbf{x}_{i'} \rangle$, $\Omega(i, i') = \langle \mathbf{z}_j, \mathbf{z}_{j'} \rangle$, and $C(i, j) = \langle \mathbf{x}_i, \mathbf{x}_j \rangle$, with $\mathbf{x} \in \mathbb{R}^{L1}$ and $\mathbf{z} \in \mathbb{R}^{L2}$. The edge-wise covariance is then

- Bipartite Graphs: $K\left((i,j),(i',j')\right) = \langle \mathbf{x}_i \otimes \mathbf{z}_j, \mathbf{x}_{i'} \otimes \mathbf{z}_{j'} \rangle$.
- Directed Graphs: $K\left((i,j),(i',j')\right) = \langle \mathbf{x}_i \otimes \mathbf{x}_j, \mathbf{x}_{i'} \otimes \mathbf{x}_{j'} \rangle$.
- Undirected Graphs: $K\left((i,j),(i',j')\right) = \langle \mathbf{x}_i \otimes \mathbf{x}_j, \mathbf{x}_{i'} \otimes \mathbf{x}_{j'} \rangle + \langle \mathbf{x}_i \otimes \mathbf{x}_j, \mathbf{x}_{j'} \otimes \mathbf{x}_{i'} \rangle$

We turn the problem of optimizing $K$ into the problem of optimizing $\mathbf{X} = [\mathbf{x}_1, \ldots, \mathbf{x}_M]^\top$ and $\mathbf{Z} = [\mathbf{z}_1, \ldots, \mathbf{z}_N]^\top$. It is important to note that in all the cases the kernel matrix has the form $\mathbf{K} = \mathbf{U}\mathbf{U}^\top$, where $\mathbf{U}$ is an $|\mathbb{O}| \times L$ matrix, $L \ll |\mathbb{O}|$, therefore applying the Woodbury identity $\mathbf{C}^{-1} = \beta[\mathbf{I} - \mathbf{U}(\mathbf{U}^\top\mathbf{U} + \beta^{-1}\mathbf{I})^{-1}\mathbf{U}^\top]$ can dramatically reduce the computational cost. For example, in the bipartite graph case and the directed graph case, respectively there are

$$\mathbf{U}^\top = \left[\mathbf{x}_i \otimes \mathbf{z}_j\right]_{(i,j)\in\mathbb{O}}, \quad \text{and} \quad \mathbf{U}^\top = \left[\mathbf{x}_i \otimes \mathbf{x}_j\right]_{(i,j)\in\mathbb{O}}, \tag{10}$$

where the rows of $\mathbf{U}$ are indexed by $(i, j) \in \mathbb{O}$. For the undirected graph case, we first rewrite the kernel function

$$K\left((i,j),(i',j')\right) = \langle \mathbf{x}_i \otimes \mathbf{x}_j, \mathbf{x}_{i'} \otimes \mathbf{x}_{j'} \rangle + \langle \mathbf{x}_i \otimes \mathbf{x}_j, \mathbf{x}_{j'} \otimes \mathbf{x}_{i'} \rangle$$
$$= \frac{1}{2}\left[\langle \mathbf{x}_i \otimes \mathbf{x}_j, \mathbf{x}_{i'} \otimes \mathbf{x}_{j'} \rangle + \langle \mathbf{x}_j \otimes \mathbf{x}_i, \mathbf{x}_{j'} \otimes \mathbf{x}_{i'} \rangle + \langle \mathbf{x}_i \otimes \mathbf{x}_j, \mathbf{x}_{j'} \otimes \mathbf{x}_{i'} \rangle + \langle \mathbf{x}_j \otimes \mathbf{x}_i, \mathbf{x}_{i'} \otimes \mathbf{x}_{j'} \rangle\right]$$
$$= \frac{1}{2}\left[\langle(\mathbf{x}_i \otimes \mathbf{x}_j + \mathbf{x}_j \otimes \mathbf{x}_i),(\mathbf{x}_{i'} \otimes \mathbf{x}_{j'} + \mathbf{x}_{j'} \otimes \mathbf{x}_{i'})\rangle\right], \tag{11}$$

and then obtain a simple form for the undirected graph case

$$\mathbf{U}^\top = \frac{1}{\sqrt{2}}\left[\mathbf{x}_i \otimes \mathbf{x}_j + \mathbf{x}_j \otimes \mathbf{x}_i\right]_{(i,j)\in\mathbb{O}} \tag{12}$$

The overall computational cost is at $\mathcal{O}(L^3 + |\mathbb{O}|L^2)$. Empirically we found that the algorithm is efficient to handle $L = 500$ when $|\mathbb{O}|$ is about millions. The gradients with respect to $\mathbf{U}$ can be found in [12]. Further calculation of gradients with respect to $\mathbf{X}$ and $\mathbf{Z}$ can be easily derived. Here we omit the details for saving the space. Finally, in order to predict the missing measurements, we only need to estimate a simple linear model $f(i, j) = \mathbf{w}^\top\mathbf{u}_{i,j} + b_{i,j}$.

## 3.1 Incorporating Additional Attributes and Learning from Discrete Observations

There are different ways to incorporate node or edge attributes into our model. A common practice is to let the kernel $K$, $\Sigma$, or $\Omega$ be some parametric function of attributes. One such choice is the RBF function. However, node or edge attributes are typically local information while the network itself is rather a global dependence structure, thus the network data often has a large part of patterns that are independent of those known predictors. In the following, via the example of placing a Bayesian prior on $\Sigma : \mathcal{U} \times \mathcal{U} \to \mathbb{R}$, we describe a flexible solution to incorporate additional knowledge. Let $\Sigma_0$ be the covariance that we wish $\Sigma$ to be apriori close to. We apply the prior $p(\Sigma) = \frac{1}{Z}\exp(-\tau E(\Sigma))$ and use its negative log-likelihood as a regularization for $\Sigma$:

$$\ell(\Sigma) = \tau E(\Sigma) = \frac{\tau}{2}\left[\log|\mathbf{\Sigma} + \gamma^{-1}\mathbf{I}| + \mathrm{tr}\left((\mathbf{\Sigma} + \gamma^{-1}\mathbf{I})^{-1}\mathbf{\Sigma}_0\right)\right] \tag{13}$$

where $\tau$ is a hyperparameter predetermined on validation data, and $\gamma^{-1}$ is a small number to be optimized. The energy function $E(\Sigma)$ is related to the KL divergence $D_{KL}(\mathcal{GP}(0, \Sigma_0) || \mathcal{GP}(0, \Sigma + \gamma^{-1}\delta))$, where $\delta(\cdot, \cdot)$ is the dirac kernel. If we let $\Sigma_0$ be the linear kernel of attributes, normalized by the dimensionality, then $E(\Sigma)$ can be derived from a likelihood of $\Sigma$ as if each dimension of the attributes is a random sample from $\mathcal{GP}(0, \Sigma + \gamma^{-1}\delta)$. If the attributes are nonlinear predictors we can conveniently set $\Sigma_0$ by a nonlinear kernel. We set $\mathbf{\Sigma_0} = \mathbf{I}$ if the corresponding attributes are absent. $\ell(\Omega)$, $\ell(C)$ and $\ell(K)$ can be set in the same way.

The observations can be discrete variables rather than real values. In this case, an appropriate likelihood function can be devised accordingly. For example, the *probit* function could be employed as the likelihood function for binary labels, which relates $f(i, j)$ to the target $y_{i,j} \in \{-1, +1\}$, by a cumulative normal $\Phi(y_{i,j}(f(i, j) + b_{i,j}))$. To preserve computationally tractability, a family of inference techniques, e.g. Laplace approximation, can be applied to finding a Gaussian distribution that approximates the true likelihood. Then, the marginal likelihood (8) can be written as an explicit expression and the gradient can be derived analytically as well.

## 4    Discussions on Related Work

**Transfer Learning**: As we have suggested before, the link prediction for bipartite graphs has a tight connection to transfer learning. To make it clear, let $f_j(\cdot) = f(\cdot, j)$, then the edge-wise function $f : \mathcal{U} \times \mathcal{V} \to \mathbb{R}$ consists of $N$ node-wise functions $f_j : \mathcal{U} \to \mathbb{R}$ for $j = 1, \ldots, N$. If we fix $\Omega(j, j') \equiv \delta(j, j')$, namely a Dirac delta function, then $f_j$ are assumed to be i.i.d. GP functions from $\mathcal{GP}(0, \Sigma)$, where each function corresponds to one learning task. This is the hierarchical Baysian model that assumes multiple tasks sharing the same GP prior [18]. In particular, the negative logarithm of $p(\{y_{i,j}\}, \{f_j\}|\Sigma)$ is

$$\mathcal{L}\big(\{f_j\}, \Sigma\big) = \sum_{j=1}^{N} \left[ \sum_{i \in \mathbb{O}_j} l\big(y_{i,j}, f_j(i)\big) + \frac{1}{2}\mathbf{f}_j \mathbf{\Sigma}^{-1} \mathbf{f}_j \right] + \frac{N}{2} \log |\mathbf{\Sigma}|, \qquad (14)$$

where $l(y_{i,j}, f_j(i)) = -\log p(y_{i,j}|f_j(i))$. The form is close to the recent convex multi-task learning in a regularization framework [3], if the log-determinant term is replaced by a trace regularization term $\lambda \text{tr}(\mathbf{\Sigma})$. It was proven in [3] that if $l(\cdot, \cdot)$ is convex with $f_j$, then the minimization of (14) is convex with jointly $\{f_j\}$ and $\Sigma$. The GP approach differs from the regularization approach in two aspects: (1) $f_j$ are treated as random variables which are marginalized out, thus we only need to estimate $\Sigma$; (2) The regularization for $\Sigma$ is a non-convex log-determinant term. Interestingly, because $\log |\mathbf{\Sigma}| \le \text{tr}(\mathbf{\Sigma}) - M$, the trace norm is the convex envelope for the log-determinant, and thus the two minimization problems are somehow doing similar things. However, the framework introduced in this paper goes beyond the two methods by introducing an informative kernel $\Omega$ between tasks. From a probabilistic modeling point of view, the independence of $\{f_j\}$ conditioned on $\Sigma$ is a restrictive assumption and even incorrect when some task-specific attributes are given (which means that $\{f_j\}$ are not exchangeable anymore). The task-specific kernel for transfer learning has been recently introduced in [4], which however increased the computational complexity by a factor of $N^2$. One contribution of this paper on transfer learning is an algorithm that can efficiently solve the learning problem with both data kernel $\Sigma$ and task kernel $\Omega$.

**Gaussian Process Latent-Variable Model (GPLVM)**: Our learning algorithm is also a generalization of GPLVM. If we enforce $\Omega(j, j') = \delta(j, j')$ in the model of bipartite graphs, then the evidence Eq. (9) is equivalent to the form of GPLVM,

$$\mathcal{L}(\Sigma, \beta) = \frac{MN}{2} \log 2\pi + \frac{N}{2} \ln |(\mathbf{\Sigma} + \beta^{-1}\mathbf{I})| + \frac{1}{2}\text{tr}\left[(\mathbf{\Sigma} + \beta^{-1}\mathbf{I})^{-1}\mathbf{Y}\mathbf{Y}^{\top}\right], \qquad (15)$$

where $\mathbf{Y}$ is a fully observed $M \times N$ matrix, the mean $\mathbf{B} = 0$, and there is no further regularization on $\Sigma$. GPLVM assumes that columns of $\mathbf{Y}$ are conditionally independent given $\Sigma$. In this paper we consider a situation with complex dependence of edges in network graphs.

**Other Related Work**: Getoor et al. [7] introduced link uncertainty in the framework of probabilistic relational models. Latent-class relational models [17, 11, 1] have been popular, aiming to find the block structure of links. Link prediction was casted as structured-output prediction in [15, 2]. Statistical models based on matrix factorization was studied by [8]. Our work is similar to [8] in the

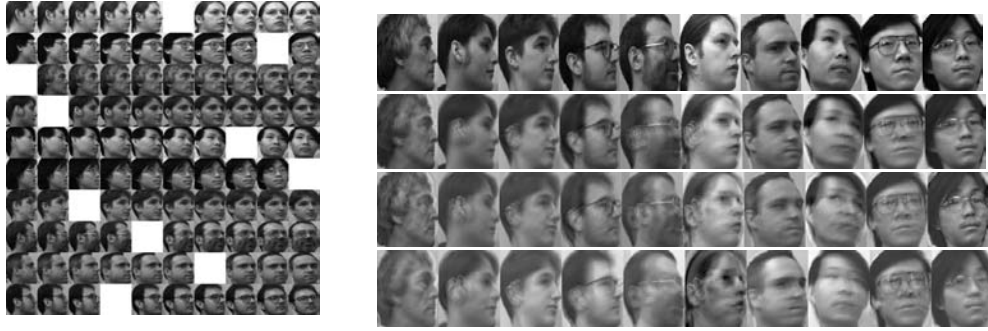

Figure 1: The left-hand side: the subset of the UMist Faces data that contains 10 people at 10 different views. The blank blocks indicate the ten knocked-off images as test cases; The right-hand side: the ten knocked-off images (the first row) along with predictive images. The second row is of our results, the third row is of the MMMF results, and the fourth row is of the bilinear results.

sense that relations are modeled by multiplications of node-wise factors. Very recently, Hoff showed in [9] that the multiplicative model generalizes the latent-class models [11, 1] and can encode the transitivity of relations.

## 5 Numerical Experiments

We set the dimensionality of the model via validation on $10\%$ of training data. In cases that the additional attributes on nodes or edges are either unavailable or very weak, we compare our method with max-margin matrix factorization (MMMF) [14] using a square loss, which is similar to singular value decomposition (SVD) but can handle missing measurements.

### 5.1 A Demonstration on Face Reconstruction

A subset of the UMist Faces images of size $112 \times 92$ was selected to illustrate our algorithm, which consists of 10 people at 10 different views. We manually knocked 10 images off as test cases, as presented in Figure 1, and treated each image as a vector that leads to a $103040 \times 10$ matrix with 103040 missing values, where each column corresponds a view of faces. GP was trained by setting $L1 = L2 = 4$ on this matrix to learn from the appearance relationships between person identity and pose. The images recovered by GP for the test cases are presented as the second row of Figure 1-right (RMSE=0.2881). The results of MMMF are presented as the third row (RMSE=0.4351). We also employed the bilinear models introduced by [16], which however does not handle missing data of a matrix, and put the results at the bottom row for comparison. Quantitatively and perceptually our model offers a better generalization to unseen views of known persons.

### 5.2 Collaborative Filtering

Collaborative filtering is a typical case of bipartite graphs, where ratings are measurements on edges of user-item pairs. We carried out a serial of experiments on the whole EachMovie data, which includes 61265 users' 2811718 distinct numeric ratings on 1623 movies. We randomly selected $80\%$ of each user's ratings for training and used the remaining $20\%$ as test cases. The random selection was carried out 20 times independently.

For comparison purpose, we also evaluated the predictive performance of four other approaches: 1) Movie Mean: the empirical mean of ratings per movie was used as the predictive value of all users' rating on the movie; 2) User Mean: the empirical mean of ratings per user was used as the predictive value of the users' rating on all movies; 3) Pearson Score: the Pearson correlation coefficient corresponds to a dot product between normalized rating vectors. We computed the Gram matrices of the Pearson score with mean imputation for movies and users respectively, and took principal components as their individual attributes. We tried 20 or 50 principal components as attributes in this experiment and carried out least square regression on observed entries. 4) MMMF. The optimal rank was decided by validation.

Table 1: Test results on the EachMovie data. The number in bracket indicates the rank we applied. The results are averaged over 20 trials, along with the standard deviation. To evaluate accuracy, we utilize root mean squared error (RMSE), mean absolute error (MAE), and normalized mean squared error, i.e. ,the RMSE normalized by the standard deviation of observations.

| METHODS | RMSE | MAE | NMSE |
|---|---|---|---|
| MOVIE MEAN | 1.3866±0.0013 | 1.1026±0.0010 | 0.7844±0.0012 |
| USER MEAN | 1.4251±0.0011 | 1.1405±0.0009 | 0.8285±0.0008 |
| PEARSON(20) | 1.3097±0.0012 | 1.0325±0.0013 | 0.6999±0.0011 |
| PEARSON(50) | 1.3034±0.0018 | 1.0277±0.0015 | 0.6931±0.0019 |
| MMMF(3) | 1.2245±0.0503 | 0.9392±0.0246 | 0.6127±0.0516 |
| MMMF(15) | 1.1696±0.0283 | 0.8918±0.0146 | 0.5585±0.0286 |
| GP(3) | 1.1557±0.0010 | 0.8781±0.0009 | 0.5449±0.0011 |

Table 2: Test results on the Cora data. The classification accuracy rate is averaged over 5 trials, each with 4 folds for training and one fold for test.

| METHODS | DS | HA | ML | PL |
|---|---|---|---|---|
| CONTENT | 53.70±0.50 | 67.50±1.70 | 68.30±1.60 | 56.40±0.70 |
| LINK | 48.90±1.70 | 65.80±1.40 | 60.70±1.10 | 58.20±0.70 |
| PCA(50) | 61.61±1.42 | 69.36±1.36 | 70.06±0.90 | 60.26±1.16 |
| GP(50) | 62.10±0.84 | 75.40±0.80 | 78.30±0.78 | 63.25±0.60 |

The results of these approaches are reported in Table 1. The per-movie average yields much better results than the per-user average, which is consistent with the findings previously reported by [5]. The improvement is noticeable by using more components of the Pearson score, but not significant. The generalization performance of our algorithm is better than that of others. T-test showed a significant difference with p-value 0.0387 of GP over MMMF (with 15 dimensions) in terms of RMSE. It is well worth highlighting another attractiveness of our algorithm – the compact representation of factors. On the EachMovie data, there are only *three factors* that well represent thousands of items individually. We also trained MMMF with 3 factors as well. Although the three-factor solution GP found is also accessible to other models, MMMF failed to achieve comparable performance on this case (i.e., see results of MMMF(3)). In each trial, the number of training samples is around 2.25 million. Our program took about 865 seconds to accomplish 500 L-BFGS updates on all 251572 parameters using an AMD Opteron 2.6GHz processor.

## 5.3 Text Categorization based on Contents and Links

We used a part of Cora corpus including 751 papers on data structure (DS), 400 papers on hardware and architecture (HA), 1617 on machine learning (ML) and 1575 on programming language (PL). We treated the citation network as a *directed graph* and modeled the link existence as binary labels. Our model applied the probit likelihood and learned a node-wise covariance function $C$, $L = 50 \times 50$, which composes an edge-wise covariance $K$ by Eq. (2). We set the prior covariance $C_0$ by the linear kernel computed by bag-of-word content attributes. Thus the learned linear features encode both link and content information, which were then used for document classification. We compare several other methods that provide linear features for one-against-all categorization using SVM: 1) CONTENT: bag-of-words features; 2) LINK: each paper's citation list; 3) PCA: 50 components by PCA on the concatenation of bag-of-word features and citation list for each paper. We chose the dimensionality 50 for both GP and PCA, because their performances both saturated when the dimensionality exceeds 50. We reported results based on 5-fold cross validation in Table 2. GP clearly outperformed other methods in 3 out of 4 categories. The main reason we believe is that our approach models the in-bound and out-bound behaviors simultaneously for each paper .

## 6 Conclusion and Extensions

In this paper we proposed GPs for modeling data living on links of networks. We described solutions to handle directed and undirected links, as well as links connecting heterogenous nodes. This work paves a way for future extensions for learning more complex relational data. For example, we can model a network containing both directed and undirected links. Let $(i, j)$ be directed and $(i', j')$ be undirected. Based on the feature representations, Eq.(10)-right for directed links and Eq.(12) for undirected links, the covaraince is $K((i, j), (i', j')) = 1/\sqrt{2}[C(i, i')C(j, j') + C(i, j')C(j, i')]$,

which indicates that dependence between a directed link and an undirected link is penalized compared to dependence between two undirected links. Moreover, GPs can be employed to model multiple networks involving multiple different types of nodes. For each type, we use one node-wise covariance. Letting covariance between two different types of nodes be zero, we obtain a huge block-diagonal node-wise covariance matrix, where each block corresponds to one type of nodes. This big covariance matrix will induce the edge-wise covariance for links connecting nodes of the same or different types. In the near future it is promising to apply the model to various link prediction or network completion problems.

## References

[1] E. M. Airoldi, D. M. Blei, S. E. Fienberg, and E. P. Xing, Mixed membership stochastic block models for relational data with application to protein-protein interactions. *Biometrics Society Annual Meeting*, 2006.

[2] S. Andrews and T. Jebara, Structured Network Learning. *NIPS Workshop on Learning to Compare Examples*, 2006.

[3] A. Argyriou, T. Evgeniou, and M. Pontil. Convex multi-task feature learning. *Machine Learning*, 2007.

[4] E. V. Bonilla, F. V. Agakov, and C. K. I. Williams. Kernel multi-task learning using task-specific features. *International Conferences on Artificial Intelligence and Statistics*, 2007.

[5] J. Canny. Collaborative filtering with privacy via factor analysis. *International ACM SIGIR Conference*, 2002.

[6] W. Chu, V. Sindhwani, Z. Ghahramani, and S. S. Keerthi. Relational learning with gaussian processes. *Neural Informaiton Processing Systems 19*, 2007.

[7] L. Getoor, E. Segal, B. Taskar, and D. Koller. Probabilistic models of text and link structure for hypertext classification. *ICJAI Workshop*, 2001.

[8] P. Hoff. Multiplicative latent factor models for description and prediction of social networks. *to appear in Computational and Mathematical Organization Theory*, 2007.

[9] P. Hoff. Modeling homophily and stochastic equivalence in symmetric relational data. *to appear in Neural Informaiton Processing Systems 20*, 2007.

[10] A. Kapoor, Y. Qi, H. Ahn, and R. W. Picard. Hyperparameter and kernel learning for graph based semi-supervised classification. *Neural Informaiton Processing Systems 18*, 2006.

[11] C. Kemp, J. B. Tenenbaum, T. L. Griffiths, T. Yamada, and N. Ueda. Learning systems of concepts with an infinite relational model. *AAAI Conference on Artificial Intelligence*, 2006.

[12] N. Lawrence. Gaussian process latent variable models. *Journal of Machine Learning Research*, 2005.

[13] C. E. Rasmussen and C. K. I. Williams. *Gaussian Processes for Machine Learning*. The MIT Press, 2006.

[14] J. D. M. Rennie and N. Srebro. Fast maximum margin matrix factorization for collaborative prediction. *International Conference on Machine Learning*, 2005.

[15] B. Taskar, M. F. Wong, P. Abbeel, and D. Koller. Link prediction in relational data. *Neural Informaiton Processing Systems 16*, 2004.

[16] J. B. Tenenbaum and W. T. Freeman. Separating style and content with bilinear models. *Neural Computation*, 2000.

[17] Z. Xu, V. Tresp, K. Yu, and H.-P. Kriegel. Infinite hidden relational models. *International Conference on Uncertainty in Artificial Intelligence*, 2006.

[18] K. Yu, V. Tresp, and A. Schwaighofer. Learning Gaussian processes from multiple tasks. *International Conference on Machine Learning*, 2005.

[19] K. Yu, W. Chu, S. Yu, V. Tresp, and Z. Xu. Stochastic relational models for discriminative link prediction. *Neural Informaiton Processing Systems 19*, 2007.

[20] X. Zhu, J. Lafferty, and Z. Ghahramani. Semi-supervised learning: From gaussian fields to gaussian processes. Technical Report CMU-CS-03-175, Carnegie Mellon University, 2003.

